# Fast Sparse Gaussian Process Methods: The Informative Vector Machine

**Neil Lawrence**
University of Sheffield
211 Portobello Street
Sheffield, S1 4DP
*neil@dcs.shef.ac.uk*

**Matthias Seeger**
University of Edinburgh
5 Forrest Hill
Edinburgh, EH1 2QL
*seeger@dai.ed.ac.uk*

**Ralf Herbrich**
Microsoft Research Ltd
7 J J Thomson Avenue
Cambridge, CB3 0FB
*rherb@microsoft.com*

## Abstract

We present a framework for sparse Gaussian process (GP) methods which uses forward selection with criteria based on information-theoretic principles, previously suggested for active learning. Our goal is not only to learn $d$–sparse predictors (which can be evaluated in $O(d)$ rather than $O(n)$, $d \ll n$, $n$ the number of training points), but also to perform training under strong restrictions on time and memory requirements. The scaling of our method is at most $O(n \cdot d^2)$, and in large real-world classification experiments we show that it can match prediction performance of the popular support vector machine (SVM), yet can be significantly faster in training. In contrast to the SVM, our approximation produces estimates of predictive probabilities ('error bars'), allows for Bayesian model selection and is less complex in implementation.

## 1 Introduction

*Gaussian process (GP)* models are powerful non-parametric tools for approximate Bayesian inference and learning. In comparison with other popular nonlinear architectures, such as multi-layer perceptrons, their behavior is conceptually simpler to understand and model fitting can be achieved without resorting to non-convex optimization routines. However, their training time scaling of $O(n^3)$ and memory scaling of $O(n^2)$, where $n$ the number of training points, has hindered their more widespread use. The related, yet non-probabilistic, *support vector machine (SVM)* classifier often renders results that are comparable to GP classifiers w.r.t. prediction error at a fraction of the training cost. This is possible because many tasks can be solved satisfactorily using *sparse* representations of the data set. The SVM is triggered towards finding such representations through the use of a particular loss function[1] that encourages some degree of sparsity, *i.e.* the final predictor depends only on a fraction of training points crucial for good discrimination on the task. Here, we call these utilized points the *active set* of the sparse predictor. In case of SVM classification, the active set contains the *support vectors*, the points closest to

the decision boundary and the misclassified ones. If the active set size $d$ is much smaller than $n$, an SVM classifier can be trained in average case running time between $O(n \cdot d^2)$ and $O(n^2 \cdot d)$ with memory requirements significantly less than $n^2$. Note, however, that without any restrictions on the data distribution, $d$ can rise to $n$.

In an effort to overcome scaling problems a range of sparse GP approximations have been proposed [1, 8, 9, 10, 11]. However, none of these has fully achieved the goals of being a nontrivial approximation to a non-sparse GP model and matching the SVM w.r.t. both prediction performance and run time. The algorithm proposed here accomplishes these objectives and, as our experiments show, can even be significantly faster in training than the SVM. Furthermore, time and memory requirements may be restricted *a priori*. The potential benefits of retaining the probabilistic characteristics of the method are numerous, since hard problems, *e.g.* feature and model selection, can be dealt with using standard techniques from Bayesian learning.

Our approach builds on earlier work of Lawrence and Herbrich [2] which we extend here by considering randomized greedy selections and focusing on an alternative representation of the GP model which facilitates generalizations to settings such as regression and multi-class classification. In the next section we introduce the GP classification model and a method for approximate inference. Section 3 then contains the derivation of our fast greedy approximation and a description of the associated algorithm. In Section 4, we present large-scale experiments on the MNIST database, comparing our method directly against the SVM. Finally we close with a discussion in Section 5.

We denote vectors $\boldsymbol{g} = (g_i)_i$ and matrices $\boldsymbol{G} = (g_{i,j})_{i,j}$ in bold-face[2]. If $I, J$ are sets of row and column indices respectively, we denote the corresponding sub-matrix of $\boldsymbol{G} \in \mathbb{R}^{p,q}$ by $\boldsymbol{G}_{I,J}$, furthermore we abbreviate $\boldsymbol{G}_{I,\cdot}$ to $\boldsymbol{G}_{I,1\ldots q}$, $\boldsymbol{G}_{I,j}$ to $\boldsymbol{G}_{I,\{j\}}$, $\boldsymbol{G}_I$ to $\boldsymbol{G}_{I,I}$, *etc.* The density of the Gaussian distribution with mean $\boldsymbol{\mu}$ and covariance matrix $\boldsymbol{\Sigma}$ is denoted by $N(\cdot|\boldsymbol{\mu}, \boldsymbol{\Sigma})$. Finally, we use diag$(\cdot)$ to represent an 'overloaded' operator which extracts the diagonal elements of a matrix as a vector or produces a square matrix with diagonal elements from a given vector, all other elements 0.

## 2   Gaussian Process Classification

Assume we are given a sample $S := ((\boldsymbol{x}_1, y_1), \ldots, (\boldsymbol{x}_n, y_n))$, $\boldsymbol{x}_i \in \mathcal{X}$, $y_i \in \{-1, +1\}$, drawn independently and identically distributed (i.i.d.) from an unknown data distribution[3] $P(\boldsymbol{x}, y)$. Our goal is to estimate $P(y|\boldsymbol{x})$ for typical $\boldsymbol{x}$ or, less ambitiously, to learn a predictor $\boldsymbol{x} \to y$ with small error on future data. To model this situation, we introduce a latent variable $u \in \mathbb{R}$ separating $\boldsymbol{x}$ and $y$, and some classification noise model $P(y|u) := \Phi(y \cdot (u + b))$, where $\Phi$ is the cumulative distribution function of the standard Gaussian $N(0, 1)$, and $b \in \mathbb{R}$ is a bias parameter. From the Bayesian viewpoint, the relationship $\boldsymbol{x} \to u$ is a random process $u(\cdot)$, which, in a *Gaussian process (GP)* model, is given a GP prior with mean function 0 and covariance kernel $k(\cdot, \cdot)$. This prior encodes the belief that (before observing any data) for any finite set $X = \{\tilde{\boldsymbol{x}}_1, \ldots, \tilde{\boldsymbol{x}}_p\} \subset \mathcal{X}$, the corresponding latent outputs $(u(\tilde{\boldsymbol{x}}_1), \ldots, u(\tilde{\boldsymbol{x}}_p))^{\mathrm{T}}$ are jointly Gaussian with mean $\boldsymbol{0} \in \mathbb{R}^p$ and covariance matrix $(k(\tilde{\boldsymbol{x}}_i, \tilde{\boldsymbol{x}}_j))_{i,j} \in \mathbb{R}^{p,p}$. GP models are *non-parametric*, that is, there is in general no finite-dimensional

parametric representation for $u(\cdot)$. It is possible to write $u(\cdot)$ as linear function in some feature space $\mathcal{F}$ associated with $k$, *i.e.* $u(\boldsymbol{x}) = \boldsymbol{w}^{\mathrm{T}}\boldsymbol{\phi}(\boldsymbol{x})$, $\boldsymbol{w} \in \mathcal{F}$, in the sense that a Gaussian prior on $\boldsymbol{w}$ induces a GP distribution on the linear function $u(\cdot)$. Here, $\boldsymbol{\phi}$ is a feature map from $\mathcal{X}$ into $\mathcal{F}$, and the covariance function can be written $k(\boldsymbol{x}, \boldsymbol{x}') = \boldsymbol{\phi}(\boldsymbol{x})^{\mathrm{T}}\boldsymbol{\phi}(\boldsymbol{x}')$. This linear function view, under which predictors become separating hyper-planes in $\mathcal{F}$, is frequently used in the SVM community. However, $\mathcal{F}$ is, in general, infinite-dimensional and not uniquely determined by the kernel function $k$. We denote the sequence of latent outputs at the training points by $\boldsymbol{u} := (u(\boldsymbol{x}_1), \ldots, u(\boldsymbol{x}_n))^{\mathrm{T}} \in \mathbb{R}^n$ and the covariance or *kernel* matrix by $\boldsymbol{K} := (k(\boldsymbol{x}_i, \boldsymbol{x}_j))_{i,j} \in \mathbb{R}^{n,n}$.

The Bayesian posterior process for $u(\cdot)$ can be computed in principle using Bayes' formula. However, if the noise model $P(y|u)$ is non-Gaussian (as is the case for binary classification), it cannot be handled tractably and is usually approximated by another Gaussian process, which should ideally preserve mean and covariance function of the former. It is easy to show that this is equivalent to fitting the moments between the *finite-dimensional* (marginal) posterior $P(\boldsymbol{u}|S)$ over the training points and a Gaussian approximation $Q(\boldsymbol{u})$, because the conditional posterior $P(u(\boldsymbol{x}_*)|\boldsymbol{u}, S)$ for some non-training point $\boldsymbol{x}_*$ is identical to the conditional prior $P(u(\boldsymbol{x}_*)|\boldsymbol{u})$. In general, computing $Q$ is also infeasible, but several authors have proposed to approximate the global moment matching by iterative schemes which locally focus on one training pattern at a time [1, 4]. These schemes (at least in their simplest forms) result in a parametric form for the approximating Gaussian

$$Q(\boldsymbol{u}) \propto P(\boldsymbol{u}) \prod_{i=1}^{n} \exp\left(-\frac{p_i}{2}(u_i - m_i)^2\right). \tag{1}$$

This may be compared with the form of the true posterior $P(\boldsymbol{u}|S) \propto P(\boldsymbol{u}) \prod_{i=1}^{n} P(y_i|u_i)$ and shows that $Q(\boldsymbol{u})$ is obtained from $P(\boldsymbol{u}|S)$ by a *likelihood approximation*. Borrowing from graphical models vocabulary, the factors in (1) are called *sites*. Initially, all $p_i, m_i$ are 0, thus $Q(\boldsymbol{u}) = P(\boldsymbol{u})$. In order to update the parameters for a site $i$, we replace it in $Q(\boldsymbol{u})$ by the corresponding true likelihood factor $P(y_i|u_i)$, resulting in a non-Gaussian distribution whose mean and covariance matrix can still be computed. This allows us to approximate it by a Gaussian $Q^{\mathrm{new}}(\boldsymbol{u})$ using moment matching. The site update is called the *inclusion* of $i$ into the active set $I$. The factorized form of the likelihood implies that the new and old $Q$ differ only in the parameters $p_i, m_i$ of site $i$. This is a useful *locality property* of the scheme which is referred to as *assumed density filtering (ADF)* (e.g. [4]). The special case of ADF[4] for GP models has been proposed in [5].

## 3   Sparse Gaussian Process Classification

The simplest way to obtain a *sparse* Gaussian process classification (GPC) approximation from the ADF scheme is to leave most of the site parameters at 0, *i.e.* $p_i = 0$, $m_i = 0$ for all $i \notin I$, where $I \subset \{1, \ldots, n\}$ is the *active set*, $|I| =: d < n$. For this to succeed, it is important to choose $I$ so that the decision boundary between classes is represented essentially as accurately as if we used the whole training set. An exhaustive search over all possible subsets $I$ is, of course, intractable. Here, we follow a greedy approach suggested in [2], including new patterns one at a time into $I$. The selection of a pattern to include is made by computing a score function for

**Algorithm 1** Informative vector machine algorithm

---
**Require:** A desired sparsity $d \ll n$.
  $I = \emptyset$, $\boldsymbol{m} = \boldsymbol{0}$, $\boldsymbol{\Pi} = \mathrm{diag}(\boldsymbol{0})$, $\mathrm{diag}(\boldsymbol{A}) = \mathrm{diag}(\boldsymbol{K})$, $\boldsymbol{h} = \boldsymbol{0}$, $J = \{1, \ldots, n\}$.
  **repeat**
    **for** $j \in J$ **do**
      Compute $\Delta_j$ according to (4).
    **end for**
    $i = \mathrm{argmax}_{j \in J} \Delta_j$
    Do updates for $p_i$ and $m_i$ according to (2).
    Update matrices $\boldsymbol{L}$, $\boldsymbol{M}$, $\mathrm{diag}(\boldsymbol{A})$ and $\boldsymbol{h}$ according to (3).
    $I \leftarrow I \cup \{i\}$, $J \leftarrow J \setminus \{i\}$.
  **until** $|I| = d$

---

all points in $J = \{1, \ldots, n\} \setminus I$ (or a subset thereof) and then picking the winner. The heuristic we implement has also been considered in the context of active learning (see chapter 5 of [3]): score an example $(\boldsymbol{x}_i, y_i)$ by the decrease in entropy of $Q(\cdot)$ upon its inclusion. As a result of the locality property of ADF and the fact that $Q$ is Gaussian, it is easy to see that the entropy difference $\mathrm{H}[Q^{\mathrm{new}}] - \mathrm{H}[Q]$ is proportional to the log ratio between the variances of the marginals $Q^{\mathrm{new}}(u_i)$ and $Q(u_i)$. Thus, our heuristic (referred to as the *differential entropy score*) favors points whose inclusion leads to a large reduction in predictive (posterior) variance at the corresponding site. Whilst other selection heuristics can be argued for and utilized, it turns out that the differential entropy score together with the simple likelihood approximation in (1) leads to an extremely efficient and competitive algorithm.

In the remainder of this section, we describe our method and give a schematic algorithm. A detailed derivation and discussions of some extensions can be found in [7]. From (1) we have $Q(\cdot) = N(\cdot | \boldsymbol{h}, \boldsymbol{A})$, $\boldsymbol{A} := (\boldsymbol{K}^{-1} + \boldsymbol{\Pi})^{-1}$, $\boldsymbol{h} := \boldsymbol{A}\boldsymbol{\Pi}\boldsymbol{m}$ and $\boldsymbol{\Pi} := \mathrm{diag}(\boldsymbol{p})$. If $I$ is the current active set, then all components of $\boldsymbol{p}$ and $\boldsymbol{m}$ not in $I$ are zero, and some algebra using the Woodbury formula gives

$$\boldsymbol{A} = \boldsymbol{K} - \boldsymbol{M}^{\mathrm{T}}\boldsymbol{M}, \qquad \boldsymbol{M} = \boldsymbol{L}^{-1}\boldsymbol{\Pi}_I^{1/2}\boldsymbol{K}_{I,\cdot} \in \mathbb{R}^{d,n},$$

where $\boldsymbol{L}$ is the lower-triangular Cholesky factor of

$$\boldsymbol{B} = \boldsymbol{I} + \boldsymbol{\Pi}_I^{1/2}\boldsymbol{K}_I\boldsymbol{\Pi}_I^{1/2} \in \mathbb{R}^{d,d}.$$

In order to compute the differential entropy score for a point $j \notin I$, we have to know $a_{j,j}$ and $h_j$. Thus, when including $i$ into the active set $I$, we need to update $\mathrm{diag}(\boldsymbol{A})$ and $\boldsymbol{h}$ accordingly, which in turn requires the matrices $\boldsymbol{L}$ and $\boldsymbol{M}$ to be kept up-to-date. The update equations for $p_i$, $m_i$ are

$$p_i = \frac{\nu_i}{1 - a_{i,i}\nu_i}, \quad m_i = h_i + \frac{\alpha_i}{\nu_i}, \qquad \text{where}$$

$$z_i = \frac{y_i \cdot (h_i + b)}{\sqrt{1 + a_{i,i}}}, \quad \alpha_i = \frac{y_i \cdot N(z_i|0,1)}{\Phi(z_i)\sqrt{1 + a_{i,i}}}, \quad \nu_i = \alpha_i \left( \alpha_i + \frac{h_i + b}{1 + a_{i,i}} \right). \tag{2}$$

We then update $\boldsymbol{L} \rightarrow \boldsymbol{L}^{\mathrm{new}}$ by appending the row $(\boldsymbol{l}^{\mathrm{T}}, l)$ and $\boldsymbol{M} \rightarrow \boldsymbol{M}^{\mathrm{new}}$ by appending the row $\boldsymbol{\mu}^{\mathrm{T}}$, where

$$\boldsymbol{l} = \sqrt{p_i}\boldsymbol{M}_{\cdot,i}, \quad l = \sqrt{1 + p_i\boldsymbol{K}_{i,i} - \boldsymbol{l}^{\mathrm{T}}\boldsymbol{l}}, \quad \boldsymbol{\mu} = l^{-1}(\sqrt{p_i}\boldsymbol{K}_{\cdot,i} - \boldsymbol{M}^{\mathrm{T}}\boldsymbol{l}). \tag{3}$$

Finally, $\mathrm{diag}(\boldsymbol{A}^{\mathrm{new}}) \leftarrow \mathrm{diag}(\boldsymbol{A}) - (\mu_j^2)_j$ and $\boldsymbol{h}^{\mathrm{new}} \leftarrow \boldsymbol{h} + \alpha_i l p_i^{-1/2}\boldsymbol{\mu}$. The differential entropy score for $j \notin I$ can be computed based on the variables in (2) (with $i \rightarrow j$) as

$$\Delta_j = \frac{1}{2}\log(1 - a_{j,j}\nu_j), \tag{4}$$

which can be computed in $O(1)$, given $h_j$ and $a_{j,j}$. In Algorithm 1 we give an algorithmic version of this scheme.

Each inclusion costs $O(n \cdot d)$, dominated by the computation of $\boldsymbol{\mu}$, apart from the computation of the kernel matrix column $\boldsymbol{K}_{\cdot,i}$. Thus the total time complexity is $O(n \cdot d^2)$. The storage requirement is $O(n \cdot d)$, dominated by the buffer for $\boldsymbol{M}$. Given $\mathrm{diag}(\boldsymbol{A})$ and $\boldsymbol{h}$, the error or the expected log likelihood of the current predictor on the remaining points $J$ can be computed in $O(n)$. These scores can be used in order to decide how many points to include into the final $I$. For kernel functions with constant diagonal, our selection heuristic is constant over patterns if $I = \emptyset$, so the first (or the first few) inclusion candidate is chosen at random. After training is complete, we can predict on test points $\boldsymbol{x}_*$ by evaluating the approximate predictive distribution $Q(u_* | \boldsymbol{x}_*, S) = \int P(u_* | \boldsymbol{u}) Q(\boldsymbol{u}) \, d\boldsymbol{u} = N(u_* | \mu(\boldsymbol{x}_*), \sigma^2(\boldsymbol{x}_*))$, where

$$\mu(\boldsymbol{x}_*) = \boldsymbol{\beta}^{\mathrm{T}} \boldsymbol{k}(\boldsymbol{x}_*), \quad \sigma^2(\boldsymbol{x}_*) = k(\boldsymbol{x}_*, \boldsymbol{x}_*) - \boldsymbol{k}(\boldsymbol{x}_*)^{\mathrm{T}} \boldsymbol{\Pi}_I^{1/2} \boldsymbol{B}^{-1} \boldsymbol{\Pi}_I^{1/2} \boldsymbol{k}(\boldsymbol{x}_*), \quad (5)$$

with $\boldsymbol{\beta} := \boldsymbol{\Pi}_I^{1/2} \boldsymbol{B}^{-1} \boldsymbol{\Pi}_I^{1/2} \boldsymbol{m}_I$ and $\boldsymbol{k}(\boldsymbol{x}_*) := (k(\boldsymbol{x}_i, \boldsymbol{x}_*))_{i \in I}$. We may compute $\sigma^2(\boldsymbol{x}_*)$ using one back-substitution with the factor $\boldsymbol{L}$. The approximate predictive distribution over $y_*$ can be obtained by averaging the noise model over the Gaussian. The optimal predictor for the approximation is $\mathrm{sgn}(\mu(\boldsymbol{x}_*) + b)$, which is independent of the variance $\sigma^2(\boldsymbol{x}_*)$.

The simple scheme above employs full greedy selection over all remaining points to find the inclusion candidate. This is sensible during early inclusions, but computationally wasteful during later ones, and an important extension of the basic scheme of [2] allows for *randomized greedy selections*. To this end, we maintain a *selection index* $J \subset \{1, \ldots, n\}$ with $J \cap I = \emptyset$ at all times. Having included $i$ into $I$ we modify the selection index $J$. This means that only the components $J$ of $\mathrm{diag}(\boldsymbol{A})$ and $\boldsymbol{h}$ have to be updated, which requires only the columns $\boldsymbol{M}_{\cdot,J}$. Hence, if $J$ exhibits some inertia while moving over $\{1, \ldots, n\} \setminus I$, many of the columns of $\boldsymbol{M}$ will not have to be kept up-to-date. In our implementation, we employ a simple delayed updating scheme for the columns of $\boldsymbol{M}$ which avoids double computations (see [7] for details). After a number of initial inclusions are done using full greedy selection, we use a $J$ of fixed size $m$ together with the following modification rule: for a fraction $\tau \in (0, 1)$, retain the $\tau \cdot m$ best-scoring points in $J$, then fill it up to size $m$ by drawing at random from $\{1, \ldots, n\} \setminus (I \cup J)$.

## 4 Experiments

We now present results of experiments on the MNIST handwritten digits database[5], comparing our method against the SVM algorithm. We considered binary tasks of the form '$c$-against-rest', $c \in \{0, \ldots, 9\}$. $c$ is mapped to $+1$, all others to $-1$. We down-sampled the bitmaps to size $13 \times 13$ and split the MNIST training set into a (new) training set of size $n = 59000$ and a validation set of size 1000; the test set size is 10000. A run consisted of model selection, training and testing, and all results are averaged over 10 runs. We employed the RBF kernel $k(\boldsymbol{x}, \boldsymbol{x}') = C \exp(-(\gamma/(2 \cdot 169)) \|\boldsymbol{x} - \boldsymbol{x}'\|^2)$, $\boldsymbol{x} \in \mathbb{R}^{169}$ with hyper-parameters $C > 0$ (process variance) and $\gamma > 0$ (inverse squared length-scale). Model selection was done by minimizing validation set error, training on random training set subsets of size 5000.[6]

|  | SVM | | | | IVM | | |
|---|---|---|---|---|---|---|---|
| $c$ | $d$ | gen | time | $c$ | $d$ | gen | time |
| 0 | 1247 | 0.22 | 1281 | 0 | 1130 | **0.18** | **627** |
| 1 | 798 | **0.20** | 864 | 1 | 820 | 0.26 | **427** |
| 2 | 2240 | **0.40** | 2977 | 2 | 2150 | **0.40** | **1690** |
| 3 | 2610 | 0.41 | 3687 | 3 | 2500 | **0.39** | **2191** |
| 4 | 1826 | 0.40 | 2442 | 4 | 1740 | **0.33** | **1210** |
| 5 | 2306 | **0.29** | 2771 | 5 | 2200 | 0.32 | **1758** |
| 6 | 1331 | **0.28** | 1520 | 6 | 1270 | 0.29 | **765** |
| 7 | 1759 | 0.54 | 2251 | 7 | 1660 | **0.51** | **1110** |
| 8 | 2636 | **0.50** | 3909 | 8 | 2470 | 0.53 | **2024** |
| 9 | 2731 | 0.58 | 3469 | 9 | 2740 | **0.55** | **2444** |

Table 1: Test error rates (gen, %) and training times (time, $s$) on binary MNIST tasks. SVM: Support vector machine (SMO); $d$: average number of SVs. IVM: Sparse GPC, randomized greedy selections; $d$: final active set size. Figures are means over 10 runs.

Our goal was to compare the methods not only w.r.t. performance, but also running time. For the SVM, we chose the SMO algorithm [6] together with a fast elaborate kernel matrix cache (see [7] for details). For the IVM, we employed randomized greedy selections with fairly conservative settings.[7] Since each binary digit classification task is very unbalanced, the bias parameter $b$ in the GPC model was chosen to be non-zero. We simply fixed $b = \Phi^{-1}(r)$, where $r$ is the ratio between $+1$ and $-1$ patterns in the training set, and added a constant $v_b = 1/10$ to the kernel $k$ to account for the variance of the bias hyper-parameter. Ideally, both $b$ and $v_b$ should be chosen by model selection, but initial experiments with different values for $(b, v_b)$ exhibited no significant fluctuations in validation errors. To ensure a fair comparison, we did initial SVM runs and initialized the active set size $d$ with the average number (over 10 runs) of SVs found, independently for each $c$. We then re-ran the SVM experiments, allowing for $O(d\,n)$ cache space. Table 1 shows the results.

Note that IVM shows comparable performance to the SVM, while achieving significantly lower training times. For less conservative settings of the randomized selection parameters, further speed-ups might be realizable. We also registered (not shown here) significant fluctuations in training time for the SVM runs, while this figure is stable and a-priori predictable for the IVM. Within the IVM, we can obtain estimates of predictive probabilities for test points, quantifying prediction uncertainties. In Figure 1, which was produced for the hardest task $c = 9$, we reject fractions of test set examples based on the size of $|P(y_* = +1) - 1/2|$. For the SVM, the size of the discriminant output is often used to quantify predictive uncertainty heuristically. For $c = 9$, the latter is clearly inferior (although the difference is less pronounced for the simpler binary tasks).

In the SVM community it is common to combine the '$c$-against-rest' classifiers to obtain a multi-class discriminant[8] as follows: for a test point $\boldsymbol{x}_*$, decide for the class whose associated classifier has the highest real-valued output. For the IVM, the

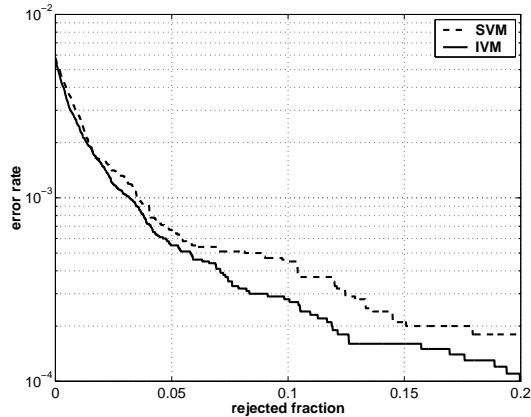

Figure 1: Plot of test error rate against increasing rejection rate for the SVM (dashed) and IVM (solid), for the task $c = 9$ against the rest. For SVM, we reject based on "distance" from separating plane, for IVM based on estimates of predictive probabilities. The IVM line runs below the SVM line exhibiting lower classification errors for identical rejection rates.

equivalent would be to compare the estimates $\log P(y_* = +1)$ from each $c$-predictor and pick the maximizing $c$. This is suboptimal, because the different predictors have not been trained jointly.[9] However, the estimates of $\log P(y_* = +1)$ do depend on *predictive variances*, *i.e.* a measure of uncertainty about the predictive mean, which cannot be properly obtained within the SVM framework. This combination scheme results in test errors of $1.54\%(\pm 0.0417\%)$ for IVM, $1.62\%(\pm 0.0316\%)$ for SVM. When comparing these results to others in the literature, recall that our experiments were based on images sub-sampled to size $13 \times 13$ rather than the usual $28 \times 28$.

## 5 Discussion

We have demonstrated that sparse Gaussian process classifiers can be constructed efficiently using greedy selection with a simple fast selection criterion. Although we focused on the change in differential entropy in our experiments here, the simple likelihood approximation at the basis of our method allows for other equally efficient criteria such as information gain [3]. Our method retains many of the benefits of probabilistic GP models (error bars, model combination, interpretability, *etc.*) while being much faster and more memory-efficient both in training and prediction. In comparison with non-probabilistic SVM classification, our method enjoys the further advantages of being simpler to implement and having strictly predictable time requirements. Our method can also be significantly faster[10] than SVM with the SMO algorithm. This is due to the fact that SMO's active set typically fluctuates heavily across the training set, thus a large fraction of the full kernel matrix must be evaluated. In contrast, IVM requires only $d/n$ of $\boldsymbol{K}$.

Among the many proposed sparse GP approximations [1, 8, 9, 10, 11], our method is most closely related to [1]. The latter is a sparse Bayesian *online* scheme which does not employ greedy selections and uses a more accurate likelihood approximation than we do, at the expense of slightly worse training time scaling, especially when compared with our randomized version. It also requires the specification of a rejection threshold and is dependent on the ordering in which the training points are presented. It incorporates steps to remove points from $I$, which can also be done straightforwardly in our scheme, however such moves are likely to create numerical stability problems. Smola and Bartlett [8] use a likelihood approximation different from both the IVM and the scheme of [1] for GP regression, together with greedy selections, but in contrast to our work they use a very expensive selection heuristic ($O(n \cdot d)$ per score computation) and are forced to use randomized greedy selection over small selection indexes. The differential entropy score has previously been suggested in the context of active learning (*e.g.* [3]), but applies more directly to our problem. In active learning, the label $y_i$ is not known at the time $\boldsymbol{x}_i$ has to be scored, and expected rather than actual entropy changes have to be considered. Furthermore, MacKay [3] applies the selection to multi-layer perceptron (MLP) models for which Gaussian posterior approximations over the weights can be very poor.

### Acknowledgments

We thank Chris Williams, David MacKay, Manfred Opper and Lehel Csató for helpful discussions. MS gratefully acknowledges support through a research studentship from *Microsoft Research Ltd*.

## Footnotes

[1] An SVM classifier is trained by minimizing a regularized loss functional, a process which cannot be interpreted as approximation to Bayesian inference.

[2]Whenever we use a bold symbol $\boldsymbol{g}$ or $\boldsymbol{G}$ for a vector or matrix, we denote its components by the corresponding normal symbols $g_i$ and $g_{i,j}$.

[3]We focus on binary classification, but our framework can be applied straightforwardly to regression estimation and multi-class classification.

[4]A generalization of ADF, *expectation propagation (EP)* [4], allows for several iterations over the data. In the context of sparse approximations, it allows us to remove points from $I$ or exchange them against such outside $I$, although we do not consider such moves here.

[5]Available online at *http://www.research.att.com/~yann/exdb/mnist/index.html*.

[6]The model selection training set for a run $i$ is the same across tested methods. The list of kernel parameters considered for selection has the same size across methods.

[7]First 2 selections at random, then 198 using full greedy, after that a selection index of size 500 and a retained fraction $\tau = 1/2$.

[8]Although much recent work has looked into more powerful combination schemes, *e.g.* based on error-correcting codes.

[9]It is straightforward to obtain the IVM for a joint GP classification model, however the training costs raise by a factor of $c^2$. Whether this factor can be reduced to $c$ using further sensible approximations, is an open question.

[10]We would expect SVMs to catch up with IVMs on tasks which require fairly large active sets, and for which very simple and fast covariance functions are appropriate (*e.g.* sparse input patterns).

### References

[1] Lehel Csató and Manfred Opper. Sparse online Gaussian processes. *N. Comp.*, 14:641–668, 2002.

[2] Neil D. Lawrence and Ralf Herbrich. A sparse Bayesian compression scheme - the informative vector machine. Presented at NIPS 2001 Workshop on Kernel Methods, 2001.

[3] David MacKay. *Bayesian Methods for Adaptive Models*. PhD thesis, California Institute of Technology, 1991.

[4] Thomas Minka. *A Family of Algorithms for Approximate Bayesian Inference*. PhD thesis, MIT, January 2001.

[5] Manfred Opper and Ole Winther. Gaussian processes for classification: Mean field algorithms. *N. Comp.*, 12(11):2655–2684, 2000.

[6] John C. Platt. Fast training of support vector machines using sequential minimal optimization. In Schölkopf et. al., editor, *Advances in Kernel Methods*, pages 185–208. 1998.

[7] Matthias Seeger, Neil D. Lawrence, and Ralf Herbrich. Sparse Bayesian learning: The informative vector machine. Technical report, Department of Computer Science, Sheffield, UK, 2002. See `www.dcs.shef.ac.uk/~neil/papers/`.

[8] Alex Smola and Peter Bartlett. Sparse greedy Gaussian process regression. In *Advances in NIPS 13*, pages 619–625, 2001.

[9] Michael Tipping. Sparse Bayesian learning and the relevance vector machine. *J. M. Learn. Res.*, 1:211–244, 2001.

[10] Volker Tresp. A Bayesian committee machine. *N. Comp.*, 12(11):2719–2741, 2000.

[11] Christopher K. I. Williams and Matthias Seeger. Using the Nyström method to speed up kernel machines. In *Advances in NIPS 13*, pages 682–688, 2001.
